# Fast Learning with Predictive Forward Models

**Carlos Brody***
Dept. of Computer Science
IIMAS, UNAM
México D.F. 01000
México.
*e-mail: carlos@hope.caltech.edu*

## Abstract

A method for transforming performance evaluation signals distal both in space and time into proximal signals usable by supervised learning algorithms, presented in [Jordan & Jacobs 90], is examined. A simple observation concerning differentiation through models trained with redundant inputs (as one of their networks is) explains a weakness in the original architecture and suggests a modification: an internal world model that encodes action-space exploration and, crucially, cancels input redundancy to the forward model is added. Learning time on an example task, cart-pole balancing, is thereby reduced about 50 to 100 times.

## 1 INTRODUCTION

In many learning control problems, the evaluation used to modify (and thus improve) control may not be available in terms of the controller's output: instead, it may be in terms of a spatial transformation of the controller's output variables (in which case we shall term it as being "distal in space"), or it may be available only several time steps into the future (termed as being "distal in time"). For example, control of a robot arm may be exerted in terms of joint angles, while evaluation may be in terms of the endpoint cartesian coordinates; furthermore, we may only wish to evaluate the endpoint coordinates reached after a certain period of time: the co-

ordinates reached at the end of some motion, for instance. In such cases, supervised learning methods are not directly applicable, and other techniques must be used. Here we study one such technique (proposed for cases where the evaluation is distal in both space and time by [Jordan & Jacobs 90]), analyse a source of its problems, and propose a simple solution for them which leads to fast, efficient learning.

We first describe two methods, and then combine them into the "predictive forward modeling" technique with which we are concerned.

## 1.1   FORWARD MODELING

"Forward Modeling" [Jordan & Rumelhart 90] is useful for dealing with evaluations which are distal in space; it involves the construction of a differentiable model to approximate the controller-action → evaluation transformation. Let our controller have internal parameters $\mathbf{w}$, output $\mathbf{c}$, and be evaluated in space $\mathbf{e}$, where $\mathbf{e} = \mathbf{e}(\mathbf{c})$ is an unknown but well-defined transformation. If there is a desired output in space $\mathbf{e}$, called $\mathbf{e}^*$, we can write an "error" function, that is, an evaluation we wish minimised, and differentiate it w.r.t. the controller's weights to obtain

$$E = (\mathbf{e}^* - \mathbf{e})^2 \qquad \frac{\partial E}{\partial \mathbf{w}} = \frac{\partial \mathbf{c}}{\partial \mathbf{w}} \cdot \frac{\partial \mathbf{e}}{\partial \mathbf{c}} \cdot \frac{\partial E}{\partial \mathbf{e}} \qquad (1)$$

Using a differentiable controller allows us to obtain the first factor in the second equation, and the third factor is also known; but the second factor is not. However, if we construct a differentiable model (called a "forward model") of $\mathbf{e}(\mathbf{c})$, then we can obtain an approximation to the second term by differentiating the model, and use this to obtain an estimate of the gradient $\partial E/\partial \mathbf{w}$ through equation (1); this can then be used for comparatively fast minimisation of $E$, and is what is known as "forward modeling".

## 1.2   PREDICTIVE CRITICS

To deal with evaluations which are distal in time, we may use a "critic" network, as in [Barto, Sutton & Anderson 83]. For a particular control policy implemented by the controller network, the critic is trained to predict the final evaluation that will be obtained given the current state – using, for example, Sutton's TD algorithm [Sutton 88]. The estimated final evaluation is then available as soon as we enter a state, and so may in turn be used to improve the control policy. This approach is closely related to dynamic programming [Barto, Sutton & Watkins 89].

## 1.3   PREDICTIVE FORWARD MODELS

While the estimated evaluation we obtain from the critic is no longer distal in time, it may still be distal in space. A natural proposal in such cases, where the evaluation signal is distal both in space and time, is to combine the two techniques described above: use a differentiable model as a predictive critic [Jordan & Jacobs 90]. If we know the desired final evaluation, we can then proceed as in equation (1) and obtain the gradient of the error w.r.t. the controller's weights. Schematically, this would look like figure 1. When using a backprop network for the predictive model,

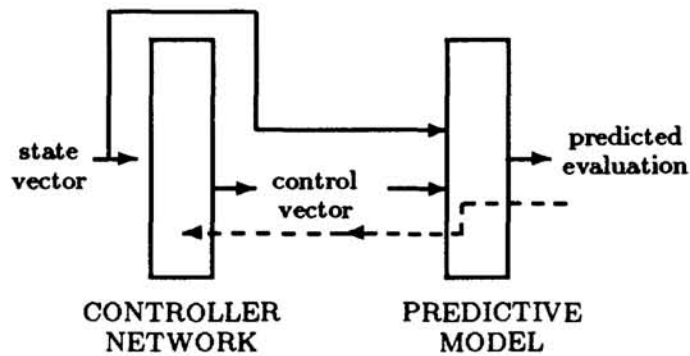

Figure 1: Jordan and Jacobs' predictive forward modeling architecture. Solid lines indicate data paths, the dashed line indicates backpropagation.

we would backpropagate through it, through it's control input, and then into the controller to modify the controller network. We should note that since predictions make no sense without a particular control policy, and the controller is only modified through the predictive model, both networks must be trained simultaneously.

[Jordan & Jacobs 90] applied this method to a well-known problem, that of learning to balance an inverted pendulum on a movable cart by exerting appropriate horizontal forces on the cart. The same task, without differentiating the critic, was studied in [Barto, Sutton & Anderson 83]. There, reinforcement learning methods were used instead to modify the controller's weights; these perform a search which in some cases may be shown to follow, *on average*, the gradient of the expected evaluation w.r.t. the network weights. Since differentiating the critic allows this gradient to be found directly, one would expect much faster learning when using the architecture of figure 1. However, Jordan and Jacobs' results show precisely the opposite: it is surprisingly slow.

## 2  THE REDUNDANCY PROBLEM

We can explain the above surprising result if we consider the fact that the predictive model network has redundant inputs: the control vector $\vec{c}$ is a function of the state vector $\vec{s}$ (call this $\vec{c} = \vec{\pi}(\vec{s})$). Let $\kappa$ and $\sigma$ be the number of components of the control and state vectors, respectively. Instead of drawing its inputs from the entire volume of $(\kappa+\sigma)$-dimensional input space, the predictor is trained **only** with inputs which lie on the $\sigma$-dimensional manifold defined by the relation $\vec{\pi}$. Away from the manifold the network is free to produce entirely arbitrary outputs. Differentiation of the model will then provide non-arbitrary gradients only for directions tangential to the manifold; this is a condition that the axes of the control dimensions will not, in general, satisfy.[1] This observation, which concerns any model trained with redundant inputs, is the very simple yet principal point of this paper.

One may argue that since the control policy is continually changing, the redundancy picture sketched out here is not in fact accurate: as the controller is modified, many

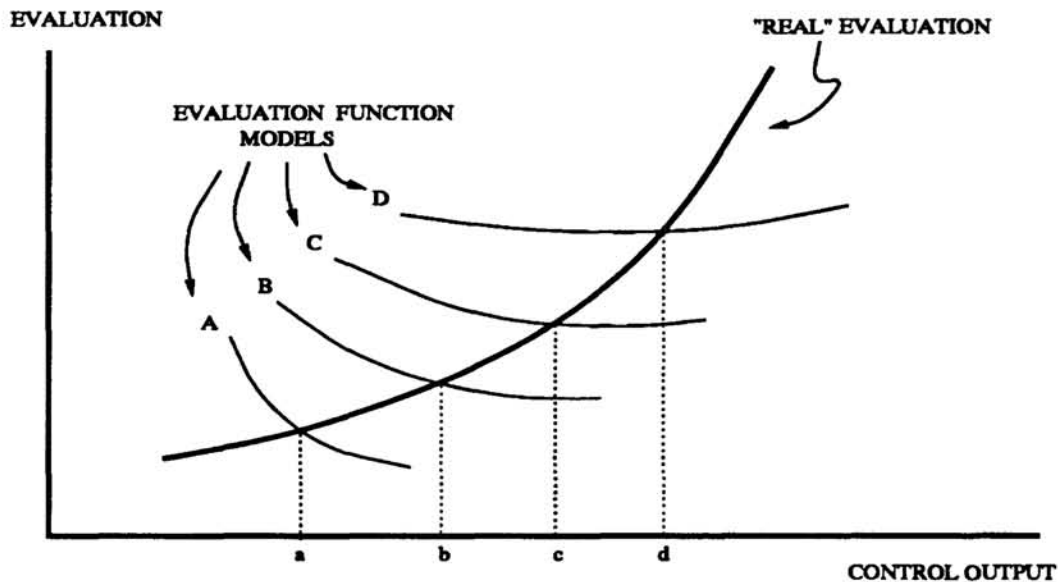

Figure 2: The evaluation as a function of control action. Curves A,B,C,D represent possible (wrong) estimates of the "real" curve made by the predictive model network.

possible control policies are "seen" by the predictor, so creating volume in input space and leading to correct gradients obtained from the predictor. However, the way in which this modification occurs is significant. An argument based on empirical observations will be made to sustain this.

Consider the example shown in figure 2. The graph shows what the "real" evaluation at some point in state space is, as a function of a component of the control action taken at that point; this function is what the predictive network should approximate. Suppose the function implemented by the predictive network initially looks like the curve which crosses the "real" evaluation function at point (a); suppose also that the current action taken also corresponds to point (a). Here we see a one-dimensional example of the redundancy problem: though the prediction at this point is entirely accurate, the gradient is not. If we wish to minimise the predicted evaluation, we would change the action in the direction of point (b). Examples of point (a) will no longer be presented to the predictive network, so it could quite plausibly modify itself simply so as to look like the estimated evaluation curve "B" which is shown crossing point (b) (a minimal change necessary to continue being correct). Again, the gradient is wrong and minimising the prediction will change the action in the same direction as before, perhaps to point (c); then to (d), and so on. Eventually, the prediction, though accurate, will have zero gradient, as in curve "D", and no modifications will occur. In practice, we have observed networks "getting stuck" in this fashion. Though the objective was to minimise the evaluation, the system stops "learning" at a point far from optimal.

The problem may be solved, as Jordan and Jacobs did, by introducing noise in the controller's output, thus breaking the redundancy. Unfortunately, this degrades

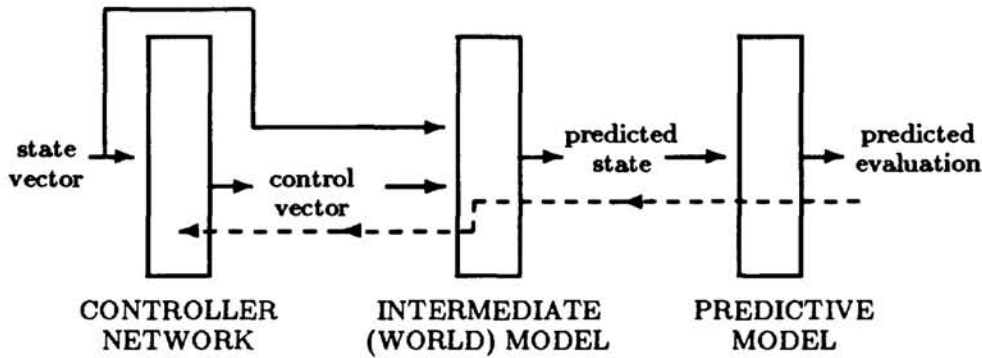

Figure 3: The proposed system architecture. Again, solid lines represent data paths while the dashed line represents backpropagation (or differentiation).

signal quality and means that since we are predicting future evaluations, we wish to predict the effects of future noise – a notoriously difficult objective. The predictive network eventually outputs the evaluation's expectation value, but this can take a long time.

# 3    USING AN INTERMEDIATE MODEL

## 3.1    AN EXTRA WORLD MODEL

Another way to solve the redundancy problem is through the use of what is here called an "intermediate model": a model of the world the controller is interacting with. That is, if $s(t)$ represents the state vector at time $t$, and $c(t)$ the controller output at time $t$, it is a model of the function $I$ where $s(t+1) = I(s(t), c(t))$.

This model is used as represented schematically in figure 3. It helps in modularising the learning task faced by the predictive model [Chrisley 90], but more interestingly, it need not be trained simultaneously with the controller since its output does not depend on future control policy. Hence, it can be trained separately, with examples drawn from its *entire* (state x action) input space, providing gradient signals without arbitrary components when differentiated. Once trained, we freeze the intermediate model's weights and insert it into the system as in figure 3; we then proceed to train the controller and predictive model as before. The predictive model will no longer have redundant inputs when trained either, so it too will provide correct gradient signals. Since all arbitrary components have been eliminated, the speedup expected from using differentiable predictive models should now be obtainable.[2]

## 3.2    AN EXAMPLE TASK

The intermediate model architecture was tested on the same example task as used by Jordan and Jacobs, that of learning to balance a pole which is attached through a hinge on its lower end to a movable cart. The control action is a real valued-force

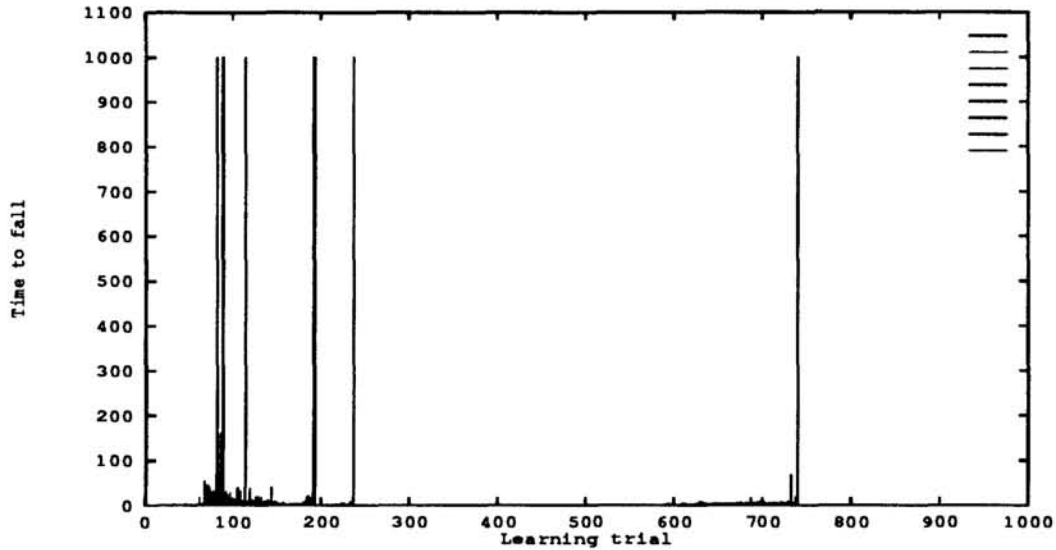

Figure 4: The evolution of eight different learning networks, using the intermediate model.

applied to the cart; the evaluation signal is a "0" while the pole has not fallen over, and the cart hasn't reached the edge of the finite-sized tracks it is allowed to move on, a "1" when either of these events happens. A trial is then said to have failed, and terminates.[3]

We count the number of learning trials needed before a controller is able to keep the pole balanced for a significant amount of a time (measured in simulated seconds). Figure 4 shows the evolution of eight networks; most reach balancing solutions within 100 to 300 faiulres. (These successful networks came from a batch of eleven: the other three never reached solutions.) This is 50 to 100 times faster than without the intermediate model, where 5000 to 30000 trials were needed to achieve similar balancing times [Jordan & Jacobs 90].

We must now take into account the overhead needed to train the intermediate model. This was done in 200 seconds of simulated time, while training the whole system typically required some 400 seconds– the overhead is small compared to the improvement achieved through the use of the intermediate model. However, off-line training of the intermediate model requires an additional agency to organise the selection and presentation of training examples. In the real world, we would either need some device which could initialise the system at any point in state space, or we would have to train through "flailing": applying random control actions, over many trials, so as to eventually cover all possible states and actions. As the dimensionality of the state representation rises for larger problems, intermediate model training will become more difficult.

## 3.3 REMARKS

We should note that the need for covering all state space is not merely due to the requirement of training an intermediate model: dynamic-programming based techniques such as the ones mentioned in this paper are guaranteed to lead us to an optimal control solution only if we explore the entire state space during learning. This is due to their generality, since no *a priori* structure of the state space is assumed. It might be possible to interleave the training of the intermediate model with the training of the controller and predictor networks, so as to achieve both concurrently. High-dimensional problems will still be problematic, but not just due to intermediate model training– the curse of dimensionality is not easily avoided!

## 4 CONCLUSIONS

If we differentiate through a model trained with redundant inputs, we eliminate possible arbitrary components (which are due to the arbitrary mixing of the inputs that the model may use) only if we differentiate tangentially along the manifold defined by the relationship between the inputs. For the architecture presented in [Jordan & Jacobs 90], this is problematic, since the axes of the control vector will typically not be tangential to the manifold. Once we take this into account, it is clear why the architecture was not as efficient as expected; and we can introduce an "intermediate" world model to avoid the problems that it had.

Using the intermediate model allows us to correctly obtain (through backpropagation, or differentiation) a real-valued vector evaluation on the controller's output. On the example task presented here, this led to a 50 to 100-fold increase in learning speed, and suggests a much better scaling-up performance and applicability to real-world problems than simple reinforcement learning, where real-valued outputs are not permitted, and vector control outputs would train very slowly.

### Acknowledgements

Many thanks are due to Richard Rohwer, who supervised the beginning of this project, and to M. I. Jordan and R. Jacobs, who answered questions enlighteningly; thanks are also due to Dr F. Bracho at IIMAS, UNAM, who provided the environment for the project's conclusion. This work was supported by scholarships from CONACYT in Mexico and from Caltech in the U.S.

## Footnotes

*Current address: Computation and Neural Systems Program, California Institute of Technology, Pasadena CA.

[1] Note that if $\vec{\pi}$ is single-valued, there is no way the manifold can "fold around" to cover all (or most) of the $\kappa + \sigma$ input space.

[2]This same architecture was independently proposed in [Werbos 90], but without the explanation as to why the intermediate model is necessary instead of merely desirable.

[3] The differential equations which were used as a model of this system may be found in [Barto, Sutton & Anderson 83]. The parameters of the simulations were identical to those used in [Jordan & Jacobs 90].

### References

[Ackley 88] D. H. Ackley, "Associative Learning via Inhibitory Search", in D. S. Touretzky, ed., *Advances in Neural Information Processing Systems 1*, Morgan Kaufmann 1989

[Barto, Sutton & Anderson 83] A. G. Barto, R. S. Sutton, and C. W. Anderson, "Neuronlike Adaptive Elements that can Solve Difficult Control Problems", IEEE Transactions on Systems, Man, and Cybernetics, Vol. SMC-13, No. 5, Sept/Oct. 1983

[Barto, Sutton & Watkins 89] A. G. Barto, R. S. Sutton, and C. J. C. H. Watkins, "Learning and Sequential Decision Making", University of Massachusetts at Amherst COINS Technical Report 89-95, September 1989

[Chrisley 90] R. L. Chrisley, "Cognitive Map Construction and Use: A Parallel Distributed Approach", in Touretzky, Elman, Sejnowski, and Hinton, eds., *Connectionist Models: Proceedings of the 1990 Summer School*, Morgan Kaufmann 1991

[Jordan & Jacobs 90] M. I. Jordan and R. A. Jacobs, "Learning to Control an Unstable System with Forward Modeling", in D. S. Touretzky, ed., *Advances in Neural Information Processing Systems 2*, Morgan Kaufmann 1990

[Jordan & Rumelhart 90] M. I. Jordan and D. E. Rumelhart, "Supervised learning with a Distal Teacher", preprint.

[Nguyen & Widrow 90] D. Nguyen and B. Widrow, "The Truck Backer-Upper: An Example of Self-Learning in Neural Networks", in Miller, Sutton and Werbos, eds., *Neural Networks for Control*, MIT Press 1990

[Sutton 88] R. S. Sutton, "Learning to Predict by the Methods of Temporal Differences", Machine Learning **3**: 9–44, 1988

[Werbos 90] P. Werbos, "Architectures for Reinforcement Learning", in Miller, Sutton and Werbos, eds., *Neural Networks for Control*, MIT Press 1990